# Bayesian morphometry of hippocampal cells suggests same-cell somatodendritic repulsion

**Giorgio A. Ascoli** [*]                                    **Alexei Samsonovich**
Krasnow Institute for Advanced Study at George Mason University
Fairfax, VA 22030-4444
*ascoli@gmu.edu*                                    *asamsono@gmu.edu*

## Abstract

Visual inspection of neurons suggests that dendritic orientation may be determined both by internal constraints (e.g. membrane tension) and by external vector fields (e.g. neurotrophic gradients). For example, basal dendrites of pyramidal cells appear nicely fan-out. This regular orientation is hard to justify completely with a general tendency to grow straight, given the zigzags observed experimentally. Instead, dendrites could (*A*) favor a fixed ("external") direction, or (*B*) repel from their own soma. To investigate these possibilities quantitatively, reconstructed hippocampal cells were subjected to Bayesian analysis. The statistical model combined linearly factors *A* and *B*, as well as the tendency to grow straight. For all morphological classes, *B* was found to be significantly positive and consistently greater than *A*. In addition, when dendrites were artificially re-oriented according to this model, the resulting structures closely resembled real morphologies. These results suggest that somatodendritic repulsion may play a role in determining dendritic orientation. Since hippocampal cells are very densely packed and their dendritic trees highly overlap, the repulsion must be cell-specific. We discuss possible mechanisms underlying such specificity.

## 1 Introduction

The study of brain dynamics and development at the cellular level would greatly benefit from a standardized, accurate and yet succinct statistical model characterizing the morphology of major neuronal classes. Such model could also provide a basis for simulation of anatomically realistic virtual neurons [1]. The model should accurately distinguish among different neuronal classes: a morphological difference between classes would be captured by a difference in model parameters and reproduced in generated virtual neurons. In addition, the model should be self-consistent: there should be no statistical difference in model parameters measured from real neurons of a given class and from virtual neurons of the same class. The assumption that a simple statistical model of this sort exists relies on the similarity of average environmental and homeostatic conditions encountered by individual neurons during development and on the limited amount of genetic information that underlies differentiation of neuronal classes.

Previous research in computational neuroanatomy has mainly focused on the topology and internal geometry of dendrites (i.e., the properties described in "dendrograms") [2,3]. Recently, we attempted to include spatial orientation in the models, thus generating

virtual neurons in 3D [4]. Dendritic growth was assumed to deviate from the straight direction both randomly and based on a constant bias in a given direction, or 'tropism'. Different models of tropism (e.g. along a fixed axis, towards a plane, or away from the soma) had dramatic effects on the shape of virtual neurons [5]. Our current strategy is to split the problem of finding a statistical model describing neuronal morphology in two parts. First, we maintain that the topology and the internal geometry of a particular dendritic tree can be described independently of its 3D embedding (i.e., the set of local dendritic orientations). At the same time, one and the same internal geometry (e.g., the experimental dendrograms obtained from real neurons) may have many equally plausible 3D embeddings that are statistically consistent with the anatomical characteristics of that neuronal class. The present work aims at finding a minimal statistical model describing local dendritic orientation in experimentally reconstructed hippocampal principal cells.

Hippocampal neurons have a polarized shape: their dendrites tend to grow from the soma as if enclosed in cones. In pyramidal cells, basal and apical dendrites invade opposite hemispaces (fig. 1A), while granule cell dendrites all invade the same hemispace. This behavior could be caused by a tendency to grow towards the layers of incoming fibers to establish synapses. Such tendency would correspond to a tropism in a direction roughly parallel to the cell main axis. Alternatively, dendrites could initially stem in the appropriate (possibly genetically determined) directions, and then continue to grow approximately in a radial direction from the soma. A close inspection of pyramidal (basal) trees suggests that dendrites may indeed be repelled from their soma (Fig. 1B). A typical dendrite may reorient itself (arrow) to grow nearly straight along a radius from the soma. Remarkably, this happens even after many turns, when the initial direction is lost. Such behavior may be hard to explain without tropism. If the deviations from straight growth were random, one should be able to 'remodel' the trees by measuring and reproducing the statistics of local turn angles, assuming its independence of dendritic orientation and location. Figure 1C shows the cell from 1A after such remodeling. In this case basal and apical dendrites retain only their initial (stemming) orientations from the original data. The resulting "cotton balls" suggests that dendritic turns are not in dependent of dendritic orientation. In this paper, we use Bayesian analysis to quantify the dendritic tropism.

## 2 Methods

Digital files of fully reconstructed rat hippocampal pyramidal cells (24 CA3 and 23 CA1 neurons) were kindly provided by Dr. D. Amaral. The overall morphology of these cells, as well as the experimental acquisition methods, were extensively described [6]. In these files, dendrites are represented as (branching) chains of cylindrical sections. Each section is connected to one other section in the path to the soma, and may be connected on the other extremity to two other sections (bifurcation), one other section (continuation point), or no other section (terminal tip). Each section is described in the file by its ending point coordinates, its diameter and its "parent", i.e., the attached section in the path to the soma [5,7]. In CA3 cells, basal dendrites had an average of 687(±216) continuation points and 72(±17) bifurcations per cell, while apical dendrites had 717(±156) continuation points and 80(±21) bifurcations per cell. CA1 cells had 462(±138) continuation points and 52(±12) bifurcations (basal), 860(±188) continuation points and 120(±22) bifurcations (apical). In the present work, basal and apical trees of CA3 and CA1 pyramidal cells were treated as 4 different classes. Digital data of rat dentate granule cells [8] are kindly made available by Dr. B. Claiborne through the internet (http://cascade.utsa.edu/bjclab). Only 36 of the 42 cells in this archive were used: in 6 cases numerical processing was not accomplished due to minor inconsistencies in the data files. The 'rejected' cells were 1208875, 3319201, 411883, 411884A, 411884B, 803887B. Granule dendrites had

549(±186) continuation points and 30(±6) bifurcations per cell. Cells in these or similar formats can be rendered, rotated, and zoomed with a java applet available through the internet (www.cns.soton.ac.uk) [7].

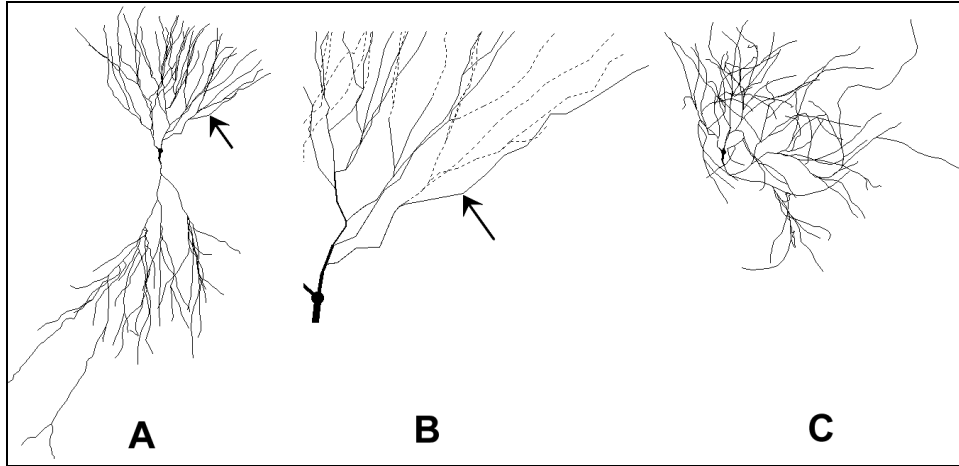

**Figure 1:** A: A pyramidal cell (c53063) from Amaral's archive. B: A zoom-in from panel A (arrows point to the same basal tree location). Dotted dendrites are behind the plane. C: Same cell (c53063) with its dendritic orientation remodeled assuming zero tropism and same statistics of all turn angles (see Results).

In agreement with the available format of morphological data (described above), the process of dendritic growth[1] can be represented as a discrete stochastic process consisting of sequential attachment of new sections to each growing dendrite. Here we keep the given internal geometry of the experimental data while remodeling the 3D embedding geometry (dendritic orientation). The task is to make a remodeled geometry statistically consistent with the original structure. The basic assumption is that neuronal development[1] is a Markov process governed by local rules [4]. Specifically, we assume that (i) each step in dendritic outgrowth only depends on the preceding step and on current local conditions; and (ii) dendrites do not undergo geometrical or topological modification after their formation (see, however, Discussion). In this Markov approximation, a plausible 3D embedding can be found by sequentially orienting individual sections, starting from the soma and moving toward the terminals. We are implementing this procedure in two-step iterations (1). First, at a given node $i$ with coordinates $\mathbf{r}_i$ we select a section $i+1$, disregard its given orientation, and calculate its most likely expected direction $\mathbf{n}'_{i+1}$ based on the model (here section $i+1$ connects nodes $i$ and $i+1$, and $\mathbf{n}$ stands for a unit vector). For a continuation point, the most likely direction $\mathbf{n}'_{i+1}$ is computed as the direction of the vector sum $\mathbf{n}_i + \mathbf{v}_i$. The first term is the direction of the parent section $\mathbf{n}_i$, and reflects the tendency dendrites exhibit to grow relatively straight due to membrane tension, mechanical properties of the cytoskeleton, etc. The second term is a local value of a vector field: $\mathbf{v}_i = \mathbf{v}(\mathbf{r}_i)$, which comprises the influence of external local conditions on the direction of growth (as specified below). Finally, we generate a perturbation of the most likely direction $\mathbf{n}'_{i+1}$ to produce a particular plausible instance of a new direction. In summary, the new direction $\mathbf{n}_{i+1}$ is generated as:

$$\mathbf{n}_{i+1} = \mathbf{T}_i\, \mathbf{n}'_{i+1}\,,$$
$$\mathbf{n}'_{i+1} \parallel \mathbf{n}_i + \mathbf{v}_i\,. \tag{1}$$

Here $\mathbf{T}_i$ is an operator that deflects $\mathbf{n}'_{i+1}$ into a random direction. If we view each deflection as a yaw of angle $\alpha_i$, then the corresponding rolling angle (describing rotation around the axis of the parent dendrite) is distributed uniformly between 0 and $2\pi$. The probability distribution function for deflections as a function of $\alpha_i$ is taken in a form that, as we found, well fits experimental data:

$$P(\mathbf{T}_i) \propto e^{-\frac{\alpha_i}{\sigma}}\,, \tag{2}$$

where $\sigma \ll 1$ is a parameter of the model. At bifurcation points, the same rule (1), (2) is applied for each daughter independently. A more accurate and plausible description of dendritic orientation at bifurcations might require a more complex model. However, our simple choice yields surprisingly good results (see below). The model (1), (2) can be used in the simulation of virtual neuronal morphology. In this case one would first need to generate the internal geometry of the dendrites [1-5]. Most importantly, model (1), (2) can be used to quantitatively assess the significance of the somatocentric (radial) tropism of real dendrites. Assuming that there is a significant preferential directionality of growth in hippocampal dendrites, the two main alternatives are (see Introduction):

**H$_A$:** The dominating tropic factor is independent of the location of the soma.
**H$_B$:** The dominating tropic factor is radial with respect to the soma.

The simplest model for the vector field $\mathbf{v}$ that discriminates between these alternative hypotheses includes both factors, A and B, linearly:

$$\mathbf{v}_i = \mathbf{a} + b\mathbf{n}_i^{\mathrm{r}}\,. \tag{3}$$

Here $\mathbf{a} = (a_x,\, a_y,\, a_z)$ is a constant vector representing global directionality of cell-independent environmental factors (chemical gradients, density of neurites, etc.) influencing dendritic orientation. $\mathbf{n}^{\mathrm{r}}_i$ is the unit vector in the direction connecting the soma to node $i$, thus representing a somatocentric tropic factor. In summary, $a_x$, $a_y$, $a_z$, $b$ and $\sigma$ are the parameters of the model. Finding that the absolute value $a = |\mathbf{a}|$ is significantly greater than $b$ would support H$_A$. On the contrary, finding that $b$ is greater than $a$ would support H$_B$. Based on a Bayesian approach, we compute the most likely values of $\mathbf{a}$, $b$ and $\sigma$ by maximization of the likelihood of all experimentally measured orientations (taken at continuation points only) of a given dendritic tree:

$$(\mathbf{a}*, b*) = \arg\max_{\{\mathbf{a},b,\sigma\}} \prod_i P(\mathbf{T}_i) = \arg\min_{\{\mathbf{a},b\}} \langle \alpha_i \rangle\,, \tag{4}$$

where $\alpha_i$ is given by (1)-(3) with experimental section orientations substituted for $\mathbf{n}_i$, $\mathbf{n}_{i+1}$, asterisk denotes most likely values, and the average is over all continuation points. Given $\mathbf{a}*$ and $b*$, the value of $\sigma*$ can be found from the average value of $\alpha_i$ computed with $\mathbf{a} = \mathbf{a}*$ and $b = b*$. The relation results from differentiation of (4) by $\sigma$. The same relation holds for the average value of $\alpha$ computed based on the probability distribution function (2) with $\sigma = \sigma*$. Therefore, $\langle\alpha_i\rangle$ computed from the neurometric data with $\mathbf{a} = \mathbf{a}*$ and $b = b*$ is equal to $\langle\alpha\rangle$ based on (2) with $\sigma = \sigma*$. The model is thus self-consistent: the measured value of $\sigma*$ in a remodeled neuron is guaranteed to coincide on average with the input parameter $\sigma$ used for simulation. In addition, our numerical analysis indicates self-consistency of the model with respect to $\mathbf{a}$ and $b$, when their values are within a practically meaningful range.

## 3 Results

Results of the Bayesian analysis are presented in Table 1. Parameters **a** and *b* were optimized for each cell individually, then the absolute value $a = |\mathbf{a}|$ was taken for each cell. The mean value and the standard deviation of *a* in Table 1 were computed based on the set of the individual absolute values, while each individual value of *b* was taken with its sign (which was positive in all cases but one). The most likely direction of **a** varied significantly among cells, i.e., no particular fixed direction was generally preferred.

**Table 1:** Results from Bayesian analysis (mean ± standard deviation). $\alpha$ is the minimized deflection angle, **a** and *b* are parameters of the model (1)-(3) computed according to (4).

| Dataset | Original data | | | Z coordinate set to zero | | |
|---|---|---|---|---|---|---|
| | $\alpha$ | *B* | *a* | $\alpha$ | *b* | *A* |
| CA3-bas | $16.4 \pm 2.3$ | $0.49 \pm 0.17$ | $0.08 \pm 0.05$ | $12.0 \pm 2.4$ | $0.42 \pm 0.15$ | $0.06 \pm 0.05$ |
| CA3-apic | $15.2 \pm 1.9$ | $0.36 \pm 0.16$ | $0.12 \pm 0.07$ | $12.0 \pm 2.9$ | $0.29 \pm 0.23$ | $0.10 \pm 0.14$ |
| CA1-bas | $16.6 \pm 1.6$ | $0.49 \pm 0.26$ | $0.14 \pm 0.10$ | $14.2 \pm 1.9$ | $0.48 \pm 0.31$ | $0.16 \pm 0.12$ |
| CA1-apic | $19.1 \pm 2.0$ | $0.30 \pm 0.20$ | $0.16 \pm 0.15$ | $17.3 \pm 2.4$ | $0.22 \pm 0.17$ | $0.11 \pm 0.10$ |
| Granule | $19.1 \pm 2.7$ | $1.01 \pm 0.64$ | $0.17 \pm 0.11$ | $11.0 \pm 1.9$ | $0.36 \pm 0.16$ | $0.07 \pm 0.05$ |

The key finding is that *a* is not significantly different from zero, while *b* is significantly positive. The slightly higher coefficient of variation obtained for granule cells could be due to a larger experimental error in the z coordinate (orthogonal to the slice). In several granule cells (but in none of the pyramidal cells) the greater noise in z was apparent upon visual inspection of the rendered structures. Therefore, we re-ran the analysis discarding the z coordinate (right columns). Results changed only minimally for pyramidal cells, and the granule cell parameters became more consistent with the pyramidal cells.

The measured average values of the model parameters were used for remodeling of experimental neuronal shapes, as described above. In particular, *b* was set to 0.5, while **a** was set to zero. We kept the internal geometry and the initial stemming direction of each tree from the experimental data, and simulated dendritic orientation at all nodes separated by more than 2 steps from the soma. A typical result is shown in Figure 2. Generally, the artificially re-oriented dendrites looked better than one could expect for a model as simple as (1) – (3). This result may be compared with figure 1C, which shows an example of remodeling based on the same model in the absence of tropism ($\mathbf{a} = b = 0$). Although in this case the shape can be improved by reducing $\sigma$, the result never gets as close to a real shape as in Fig. 2 C, D, even when random, uncorrelated local distortions ("shuffling") are applied to the generated geometry. Thus, although the tendency to grow straight represents the dominant component of the model (i.e., *b*<1), somatocentric tropism may exert a dramatic effect on dendritic shape. Surprisingly, even the asymmetry of the dendritic spread (compare front and side views) is preserved after remodeling. However, two details are difficult to reproduce with this model: the uniform distribution of dendrites in space and other subtle medium-distance correlations among dendritic deflections. In order to account for these properties, we may need to consider spatially correlated inhomogeneities of the medium and possible short range dendrodendritic interactions.

## 4 Discussion

The key results of this work is that, according to Bayesian analysis, dendrites of hippocampal principal cells display a significant radial tropism. This means that the spatial orientation of these neuronal trees can be statistically described *as if* dendrites were repelled from their own soma. This preferential direction is stronger than any tendency to grow along a fixed direction independent of the location of the soma. These results apply to all dendritic classes, but in general pyramidal cell basal trees (and granule cell dendrites) display a bigger somatocentric tropism than apical trees.

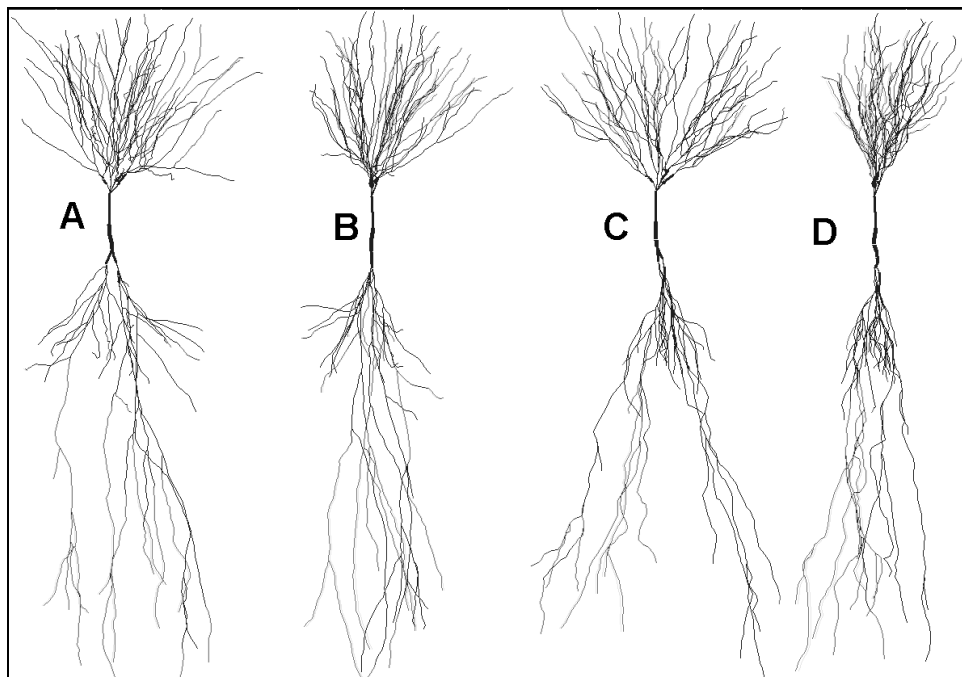

**Figure 2:** Dendritic remodeling with somatocentric tropism. A, B: front and side views of cell 10861 from Amaral' s archive. C, D: Same views after remodeling with parameters $a = 0$, $b = 0.5$, $\sigma = 0.15$ (corresponding to $<\alpha> = 17°$). The first two sections of each tree stem were taken in their original orientations; all subsequent experimental orientations were disregarded and regenerated from scratch according to the model.

Assuming that dendrites are indeed repelled from their soma during development, what could be a plausible mechanism? Principal cells are very densely packed in the hippocampus, and their dendrites highly overlap. If repulsion were mediated by a diffusible chemical factor, in order for dendrites to be repelled radially from their *own* soma, each neuron should have its own specific chemical marker (a fairly unlikely possibility). If the same repulsive factor were released by all neurons, each dendrite would be repelled by hundreds of somata, and not just by their own. The resulting tropism would be perpendicular to the principal cell layer, i.e. each dendrite would be pushed approximately in the same direction, independent of the location of its soma. This scenario is in clear contrast with the result of our statistical analysis. Thus, how can a growing dendrite sense the location of its own soma? One possibility involves the spontaneous spiking activity of neurons during development. A cell that spikes becomes unique in its neighborhood for a short period of time. The philopodia of dendritic growth

cones could possess voltage-gated receptors to sense transient chemical gradients (e.g., pH) created by the spiking cell. Only dendrites that are depolarized during the transient chemical gradient (i.e., those belonging to the same spiking cell) would be repelled by it. Alternatively, depolarized philopodia could be sensitive to the small voltage difference created by the spike in the extracellular space (a voltage that can be recorded by tetrodes).

The main results obtained with the simple model presented in this work are independent of the z coordinate in the morphometric files, i.e. the most error-prone measurement in the experimental reconstruction. However, it is important to note that any observed deviation of dendritic path from a straight line, including that due to measurement errors, causes an increase in the most likely values of parameters $a$ and $b$. Another possibility is that dendrites do grow almost precisely in straight lines, and the measured values of **a** and $b$ reflect distortions of dendritic shapes after development. In order to assess the effect of these factors on **a** and $b$, we pre-processed the experimental data by adding a gradually increasing noise to all coordinates of dendritic sections. Then we were able to extrapolate the dependence of $a^*$, $b^*$ and $<\alpha>^*$ on the amplitude of noise in order to estimate the parameter values in the absence of the experimental error (which was conservatively taken to be of 0.5 μm). For basal trees of CA3 pyramidal cells, this analysis yielded an estimated "corrected" value of $b$ between 0.14 and 0.25, with $a$ remaining much smaller than $b$. Interestingly, our analysis based on extrapolation shows that, regardless of the assumed amount of distortion present in the experimental data, given the numbers measured for CA3 basal trees, positive initial $<\alpha>$ implies positive initial $b$. In other words, not only measurement errors, but also possible biological distortions of the dendritic tree may not be capable of accounting for the observed positivity of the parameter b. Although these factors affect our results quantitatively, they do not change the statistical significance nor the qualitative trends. However, a more rigorous analysis needs to be carried out. Nevertheless, artificially reoriented dendrites according to our simple model appear almost as realistic as the original structures, and we could not achieve the same result with any choice of parameters in models of distortion without a somatocentric tropism. In conclusion, whether the present Bayesian analysis reveals a phenomenon of somatodendritic repulsion remains an (experimentally testable) open question. What is unquestionable is that the presented model is a significant step forward in the formulation of an accurate statistical description of dendritic morphology.

### Acknowledgments
This work was supported in part by Human Brain Project Grant R01 NS39600, funded jointly by NINDS and NIMH.

## Footnotes

[1] Although we resort to a developmental metaphor, our goal is to describe accurately the result of development rather than the process of development.

### References
[1] Ascoli G.A. (1999) Progress and perspectives in computational neuroanatomy. Anat. Rec. 257(6):195-207.
[2] van Pelt J. (1997) Effect of pruning on dendritic tree topology. J. Theor. Biol. 186(1):17-32.
[3] Burke R.E., W. Marks, B. Ulfhake (1992) A parsimonious description of motoneurons dendritic morphology using computer simulation. J. Neurosci. 12(6):2403-2416.
[4] Ascoli G.A., J. Krichmar (2000) L-Neuron: a modeling tool for the efficient generation and parsimonious description of dendritic morphology. Neurocomputing 32-33:1003-1011.
[5] Ascoli G.A., J. Krichmar, S. Nasuto, S. Senft (2001) Generation, description and storage of dendritic morphology data. Phil. Trans. R. Sci. B, *In Press*.
[6] Ishizuka N., W. Cowan, D. Amaral (1995) A quantitative analysis of the dendritic organization of pyramidal cells in the rat hippocampus. J. Comp. Neurol. 362(1):17-45.
[7] Cannon R.C., D. Turner, G. Pyapali, H. Wheal (1998) An on-line archive of reconstructed hippocampal neurons. J Neurosci. Meth. 84(1-2):49-54.
[8] Rihn L.L., B. Claiborne (1990) Dendritic growth and regression in rat dentate granule cells during late postnatal development. Dev. Brain Res. 54(1):115-124
